# Inter-domain Gaussian Processes for Sparse Inference using Inducing Features

**Miguel Lázaro-Gredilla and Aníbal R. Figueiras-Vidal**
Dep. Signal Processing & Communications
Universidad Carlos III de Madrid, SPAIN
{miguel,arfv}@tsc.uc3m.es

## Abstract

We present a general inference framework for inter-domain Gaussian Processes (GPs) and focus on its usefulness to build sparse GP models. The state-of-the-art sparse GP model introduced by Snelson and Ghahramani in [1] relies on finding a small, representative pseudo data set of $m$ elements (from the same domain as the $n$ available data elements) which is able to explain existing data well, and then uses it to perform inference. This reduces inference and model selection computation time from $\mathcal{O}(n^3)$ to $\mathcal{O}(m^2n)$, where $m \ll n$. Inter-domain GPs can be used to find a (possibly more compact) representative set of features lying in a different domain, at the same computational cost. Being able to specify a different domain for the representative features allows to incorporate prior knowledge about relevant characteristics of data and detaches the functional form of the covariance and basis functions. We will show how previously existing models fit into this framework and will use it to develop two new sparse GP models. Tests on large, representative regression data sets suggest that significant improvement can be achieved, while retaining computational efficiency.

## 1 Introduction and previous work

Along the past decade there has been a growing interest in the application of Gaussian Processes (GPs) to machine learning tasks. GPs are probabilistic non-parametric Bayesian models that combine a number of attractive characteristics: They achieve state-of-the-art performance on supervised learning tasks, provide probabilistic predictions, have a simple and well-founded model selection scheme, present no overfitting (since parameters are integrated out), etc.

Unfortunately, the direct application of GPs to regression problems (with which we will be concerned here) is limited due to their training time being $\mathcal{O}(n^3)$. To overcome this limitation, several sparse approximations have been proposed [2, 3, 4, 5, 6]. In most of them, sparsity is achieved by projecting all available data onto a smaller subset of size $m \ll n$ (the active set), which is selected according to some specific criterion. This reduces computation time to $\mathcal{O}(m^2n)$. However, active set selection interferes with hyperparameter learning, due to its non-smooth nature (see [1, 3]).

These proposals have been superseded by the Sparse Pseudo-inputs GP (SPGP) model, introduced in [1]. In this model, the constraint that the samples of the active set (which are called pseudo-inputs) must be selected among training data is relaxed, allowing them to lie anywhere in the input space. This allows both pseudo-inputs and hyperparameters to be selected in a joint continuous optimisation and increases flexibility, resulting in much superior performance.

In this work we introduce Inter-Domain GPs (IDGPs) as a general tool to perform inference across domains. This allows to remove the constraint that the pseudo-inputs must remain within the same domain as input data. This added flexibility results in an increased performance and allows to encode prior knowledge about other domains where data can be represented more compactly.

## 2 Review of GPs for regression

We will briefly state here the main definitions and results for regression with GPs. See [7] for a comprehensive review.

Assume we are given a training set with $n$ samples $\mathcal{D} \equiv \{\mathbf{x}_j, y_j\}_{j=1}^n$, where each $D$-dimensional input $\mathbf{x}_j$ is associated to a scalar output $y_j$. The regression task goal is, given a new input $\mathbf{x}_*$, predict the corresponding output $y_*$ based on $\mathcal{D}$.

The GP regression model assumes that the outputs can be expressed as some noiseless latent function plus independent noise, $y = f(\mathbf{x}) + \varepsilon$, and then sets a zero-mean[1] GP prior on $f(\mathbf{x})$, with covariance $k(\mathbf{x}, \mathbf{x}')$, and a zero-mean Gaussian prior on $\varepsilon$, with variance $\sigma^2$ (the noise power hyperparameter). The covariance function encodes prior knowledge about the smoothness of $f(\mathbf{x})$. The most common choice for it is the Automatic Relevance Determination Squared Exponential (ARD SE):

$$k(\mathbf{x}, \mathbf{x}') = \sigma_0^2 \exp\left[-\frac{1}{2}\sum_{d=1}^D \frac{(x_d - x_d')^2}{\ell_d^2}\right], \tag{1}$$

with hyperparameters $\sigma_0^2$ (the latent function power) and $\{\ell_d\}_{d=1}^D$ (the length-scales, defining how rapidly the covariance decays along each dimension). It is referred to as ARD SE because, when coupled with a model selection method, non-informative input dimensions can be removed automatically by growing the corresponding length-scale. The set of hyperparameters that define the GP are $\boldsymbol{\theta} = \{\sigma^2, \sigma_0^2, \{\ell_d\}_{d=1}^D\}$. We will omit the dependence on $\boldsymbol{\theta}$ for the sake of clarity.

If we evaluate the latent function at $\mathbf{X} = \{\mathbf{x}_j\}_{j=1}^n$, we obtain a set of latent variables following a joint Gaussian distribution $p(\mathbf{f}|\mathbf{X}) = \mathcal{N}(\mathbf{f}|\mathbf{0}, \mathbf{K_{ff}})$, where $[\mathbf{K_{ff}}]_{ij} = k(\mathbf{x}_i, \mathbf{x}_j)$. Using this model it is possible to express the joint distribution of training and test cases and then condition on the observed outputs to obtain the predictive distribution for any test case

$$p_{\mathrm{GP}}(y_*|\mathbf{x}_*, \mathcal{D}) = \mathcal{N}(y_*|\mathbf{k}_{\mathbf{f}*}^\top (\mathbf{K_{ff}} + \sigma^2\mathbf{I}_n)^{-1}\mathbf{y}, \ \sigma^2 + k_{**} - \mathbf{k}_{\mathbf{f}*}^\top (\mathbf{K_{ff}} + \sigma^2\mathbf{I}_n)^{-1}\mathbf{k}_{\mathbf{f}*}), \tag{2}$$

where $\mathbf{y} = [y_1, \ldots, y_n]^\top$, $\mathbf{k}_{\mathbf{f}*} = [k(\mathbf{x}_1, \mathbf{x}_*), \ldots, k(\mathbf{x}_n, \mathbf{x}_*)]^\top$, and $k_{**} = k(\mathbf{x}_*, \mathbf{x}_*)$. $\mathbf{I}_n$ is used to denote the identity matrix of size $n$. The $\mathcal{O}(n^3)$ cost of these equations arises from the inversion of the $n \times n$ covariance matrix. Predictive distributions for additional test cases take $\mathcal{O}(n^2)$ time each. These costs make standard GPs impractical for large data sets.

To select hyperparameters $\boldsymbol{\theta}$, Type-II Maximum Likelihood (ML-II) is commonly used. This amounts to selecting the hyperparameters that correspond to a (possibly local) maximum of the log-marginal likelihood, also called log-evidence.

## 3 Inter-domain GPs

In this section we will introduce Inter-Domain GPs (IDGPs) and show how they can be used as a framework for computationally efficient inference. Then we will use this framework to express two previous relevant models and develop two new ones.

### 3.1 Definition

Consider a real-valued GP $f(\mathbf{x})$ with $\mathbf{x} \in \mathbb{R}^D$ and some deterministic real function $g(\mathbf{x}, \mathbf{z})$, with $\mathbf{z} \in \mathbb{R}^H$. We define the following transformation:

$$u(\mathbf{z}) = \int_{\mathbb{R}^D} f(\mathbf{x})g(\mathbf{x}, \mathbf{z})d\mathbf{x}. \tag{3}$$

There are many examples of transformations that take on this form, the Fourier transform being one of the best known. We will discuss possible choices for $g(\mathbf{x}, \mathbf{z})$ in Section 3.3; for the moment we will deal with the general form. Since $u(\mathbf{z})$ is obtained by a linear transformation of GP $f(\mathbf{x})$,

it is also a GP. This new GP may lie in a different domain of possibly different dimension. This transformation is not invertible in general, its properties being defined by $g(\mathbf{x}, \mathbf{z})$.

IDGPs arise when we jointly consider $f(\mathbf{x})$ and $u(\mathbf{z})$ as a single, "extended" GP. The mean and covariance function of this extended GP are overloaded to accept arguments from both the input and transformed domains and treat them accordingly. We refer to each version of an overloaded function as an *instance*, which will accept a different type of arguments. If the distribution of the original GP is $f(\mathbf{x}) \sim \mathcal{GP}(m(\mathbf{x}), k(\mathbf{x}, \mathbf{x}'))$, then it is possible to compute the remaining instances that define the distribution of the extended GP over both domains. The transformed-domain instance of the mean is

$$m(\mathbf{z}) = \mathbb{E}[u(\mathbf{z})] = \int_{\mathbb{R}^D} \mathbb{E}[f(\mathbf{x})] g(\mathbf{x}, \mathbf{z}) d\mathbf{x} = \int_{\mathbb{R}^D} m(\mathbf{x}) g(\mathbf{x}, \mathbf{z}) d\mathbf{x}.$$

The inter-domain and transformed-domain instances of the covariance function are:

$$k(\mathbf{x}, \mathbf{z}') = \mathbb{E}[f(\mathbf{x}) u(\mathbf{z}')] = \mathbb{E}\left[ f(\mathbf{x}) \int_{\mathbb{R}^D} f(\mathbf{x}') g(\mathbf{x}', \mathbf{z}') d\mathbf{x}' \right] = \int_{\mathbb{R}^D} k(\mathbf{x}, \mathbf{x}') g(\mathbf{x}', \mathbf{z}') d\mathbf{x}' \quad (4)$$

$$k(\mathbf{z}, \mathbf{z}') = \mathbb{E}[u(\mathbf{z}) u(\mathbf{z}')] = \mathbb{E}\left[ \int_{\mathbb{R}^D} f(\mathbf{x}) g(\mathbf{x}, \mathbf{z}) d\mathbf{x} \int_{\mathbb{R}^D} f(\mathbf{x}') g(\mathbf{x}', \mathbf{z}') d\mathbf{x}' \right]$$

$$= \int_{\mathbb{R}^D} \int_{\mathbb{R}^D} k(\mathbf{x}, \mathbf{x}') g(\mathbf{x}, \mathbf{z}) g(\mathbf{x}', \mathbf{z}') d\mathbf{x} d\mathbf{x}'. \quad (5)$$

Mean $m(\cdot)$ and covariance function $k(\cdot, \cdot)$ are therefore defined both by the values and domains of their arguments. This can be seen as if each argument had an additional domain indicator used to select the instance. Apart from that, they define a regular GP, and all standard properties hold. In particular $k(\mathbf{a}, \mathbf{b}) = k(\mathbf{b}, \mathbf{a})$. This approach is related to [8], but here the latent space is defined as a transformation of the input space, and not the other way around. This allows to pre-specify the desired input-domain covariance. The transformation is also more general: Any $g(\mathbf{x}, \mathbf{z})$ can be used.

We can sample an IDGP at $n$ input-domain points $\mathbf{f} = [f_1,\ f_2, \ldots,\ f_n]^\top$ (with $f_j = f(\mathbf{x}_j)$) and $m$ transformed-domain points $\mathbf{u} = [u_1,\ u_2, \ldots,\ u_m]^\top$ (with $u_i = u(\mathbf{z}_i)$). With the usual assumption of $f(\mathbf{x})$ being a zero mean GP and defining $\mathbf{Z} = \{\mathbf{z}_i\}_{i=1}^m$, the joint distribution of these samples is:

$$p\left( \begin{bmatrix} \mathbf{f} \\ \mathbf{u} \end{bmatrix} \middle| \mathbf{X},\ \mathbf{Z} \right) = \mathcal{N}\left( \begin{bmatrix} \mathbf{f} \\ \mathbf{u} \end{bmatrix} \middle| \mathbf{0},\ \begin{bmatrix} \mathbf{K_{ff}} & \mathbf{K_{fu}} \\ \mathbf{K_{fu}^\top} & \mathbf{K_{uu}} \end{bmatrix} \right), \quad (6)$$

$$\text{with}\ \ [\mathbf{K_{ff}}]_{pq} = k(\mathbf{x}_p,\ \mathbf{x}_q), \quad [\mathbf{K_{fu}}]_{pq} = k(\mathbf{x}_p,\ \mathbf{z}_q), \quad [\mathbf{K_{uu}}]_{pq} = k(\mathbf{z}_p,\ \mathbf{z}_q),$$

which allows to perform inference across domains. We will only be concerned with one input domain and one transformed domain, but IDGPs can be defined for any number of domains.

## 3.2 Sparse regression using inducing features

In the standard regression setting, we are asked to perform inference about the latent function $f(\mathbf{x})$ from a data set $\mathcal{D}$ lying in the input domain. Using IDGPs, we can use data from any domain to perform inference in the input domain. Some latent functions might be better defined by a set of data lying in some transformed space rather than in the input space. This idea is used for sparse inference.

Following [1] we introduce a pseudo data set, but here we place it in the transformed domain: $\overline{\mathcal{D}} = \{\mathbf{Z}, \mathbf{u}\}$. The following derivation is analogous to that of SPGP. We will refer to $\mathbf{Z}$ as the *inducing features* and $\mathbf{u}$ as the inducing variables. The key approximation leading to sparsity is to set $m \ll n$ and assume that $f(\mathbf{x})$ is well-described by the pseudo data set $\overline{\mathcal{D}}$, so that any two samples (either from the training or test set) $f_p$ and $f_q$ with $p \neq q$ will be independent given $\mathbf{x}_p$, $\mathbf{x}_q$ and $\overline{\mathcal{D}}$. With this simplifying assumption[2], the prior over $\mathbf{f}$ can be factorised as a product of marginals:

$$p(\mathbf{f}|\mathbf{X}, \mathbf{Z}, \mathbf{u}) \approx \prod_{j=1}^n p(f_j|\mathbf{x}_j, \mathbf{Z}, \mathbf{u}). \quad (7)$$

Marginals are in turn obtained from (6): $p(f_j|\mathbf{x}_j, \mathbf{Z}, \mathbf{u}) = \mathcal{N}(f_j|\mathbf{k}_j \mathbf{K}_{\mathbf{uu}}^{-1}\mathbf{u}, \lambda_j)$, where $\mathbf{k}_j$ is the j-th row of $\mathbf{K}_{\mathbf{fu}}$ and $\lambda_j$ is the j-th element of the diagonal of matrix $\mathbf{\Lambda}_{\mathbf{f}} = \text{diag}(\mathbf{K}_{\mathbf{ff}} - \mathbf{K}_{\mathbf{fu}}\mathbf{K}_{\mathbf{uu}}^{-1}\mathbf{K}_{\mathbf{uf}})$. Operator $\text{diag}(\cdot)$ sets all off-diagonal elements to zero, so that $\mathbf{\Lambda}_{\mathbf{f}}$ is a diagonal matrix.

Since $p(\mathbf{u}|\mathbf{Z})$ is readily available and also Gaussian, the inducing variables can be integrated out from (7), yielding a new, approximate prior over $f(\mathbf{x})$:

$$p(\mathbf{f}|\mathbf{X}, \mathbf{Z}) = \int p(\mathbf{f}, \mathbf{u}|\mathbf{X}, \mathbf{Z})d\mathbf{u} \approx \int \prod_{j=1}^{n} p(f_j|\mathbf{x}_j, \mathbf{Z}, \mathbf{u})p(\mathbf{u}|\mathbf{Z})d\mathbf{u} = \mathcal{N}(\mathbf{f}|\mathbf{0}, \mathbf{K}_{\mathbf{fu}}\mathbf{K}_{\mathbf{uu}}^{-1}\mathbf{K}_{\mathbf{uf}} + \mathbf{\Lambda}_{\mathbf{f}})$$

Using this approximate prior, the posterior distribution for a test case is:

$$p_{\text{IDGP}}(y_*|\mathbf{x}_*, \mathcal{D}, \mathbf{Z}) = \mathcal{N}(y_*|\mathbf{k}_{\mathbf{u}*}^\top \mathbf{Q}^{-1}\mathbf{K}_{\mathbf{fu}}^\top\mathbf{\Lambda}_{\mathbf{y}}^{-1}\mathbf{y}, \ \sigma^2 + k_{**} + \mathbf{k}_{\mathbf{u}*}^\top(\mathbf{Q}^{-1} - \mathbf{K}_{\mathbf{uu}}^{-1})\mathbf{k}_{\mathbf{u}*}), \quad (8)$$

where we have defined $\mathbf{Q} = \mathbf{K}_{\mathbf{uu}} + \mathbf{K}_{\mathbf{fu}}^\top\mathbf{\Lambda}_{\mathbf{y}}^{-1}\mathbf{K}_{\mathbf{fu}}$ and $\mathbf{\Lambda}_{\mathbf{y}} = \mathbf{\Lambda}_{\mathbf{f}} + \sigma^2\mathbf{I}_n$. The distribution (2) is approximated by (8) with the information available in the pseudo data set. After $\mathcal{O}(m^2 n)$ time precomputations, predictive means and variances can be computed in $\mathcal{O}(m)$ and $\mathcal{O}(m^2)$ time per test case, respectively. This model is, in general, non-stationary, even when it is approximating a stationary input-domain covariance and can be interpreted as a degenerate GP plus heteroscedastic white noise.

The log-marginal likelihood (or log-evidence) of the model, explicitly including the conditioning on kernel hyperparameters $\boldsymbol{\theta}$ can be expressed as

$$\log p(\mathbf{y}|\mathbf{X}, \mathbf{Z}, \boldsymbol{\theta}) = -\frac{1}{2}[\mathbf{y}^\top\mathbf{\Lambda}_{\mathbf{y}}^{-1}\mathbf{y} - \mathbf{y}^\top\mathbf{\Lambda}_{\mathbf{y}}^{-1}\mathbf{K}_{\mathbf{fu}}\mathbf{Q}^{-1}\mathbf{K}_{\mathbf{fu}}^\top\mathbf{\Lambda}_{\mathbf{y}}^{-1}\mathbf{y} + \log(|\mathbf{Q}||\mathbf{\Lambda}_{\mathbf{y}}|/|\mathbf{K}_{\mathbf{uu}}|) + n\log(2\pi)]$$

which is also computable in $\mathcal{O}(m^2 n)$ time.

Model selection will be performed by jointly optimising the evidence with respect to the hyperparameters and the inducing features. If analytical derivatives of the covariance function are available, conjugate gradient optimisation can be used with $\mathcal{O}(m^2 n)$ cost per step.

### 3.3  On the choice of $g(\mathbf{x}, \mathbf{z})$

The *feature extraction function* $g(\mathbf{x}, \mathbf{z})$ defines the transformed domain in which the pseudo data set lies. According to (3), the inducing variables can be seen as projections of the target function $f(\mathbf{x})$ on the feature extraction function over the whole input space. Therefore, each of them summarises information about the behaviour of $f(\mathbf{x})$ everywhere. The inducing features $\mathbf{Z}$ define the concrete set of functions over which the target function will be projected. It is desirable that this set captures the most significant characteristics of the function. This can be achieved either using prior knowledge about data to select $\{g(\mathbf{x}, \mathbf{z}_i)\}_{i=1}^{m}$ or using a very general family of functions and letting model selection automatically choose the appropriate set.

Another way to choose $g(\mathbf{x}, \mathbf{z})$ relies on the form of the posterior. The posterior mean of a GP is often thought of as a linear combination of "basis functions". For full GPs and other approximations such as [1, 2, 3, 4, 5, 6], basis functions must have the form of the input-domain covariance function. When using IDGPs, basis functions have the form of the inter-domain instance of the covariance function, and can therefore be adjusted by choosing $g(\mathbf{x}, \mathbf{z})$, independently of the input-domain covariance function.

If two feature extraction functions $g(\cdot, \cdot)$ and $h(\cdot, \cdot)$ can be related by $g(\mathbf{x}, \mathbf{z}) = h(\mathbf{x}, \mathbf{z})r(\mathbf{z})$ for any function $r(\cdot)$, then both yield the same sparse GP model. This property can be used to simplify the expressions of the instances of the covariance function.

In this work we use the same functional form for every feature, i.e. our function set is $\{g(\mathbf{x}, \mathbf{z}_i)\}_{i=1}^{m}$, but it is also possible to use sets with different functional forms for each inducing feature, i.e. $\{g_i(\mathbf{x}, \mathbf{z}_i)\}_{i=1}^{m}$ where each $\mathbf{z_i}$ may even have a different size (dimension). In the sections below we will discuss different possible choices for $g(\mathbf{x}, \mathbf{z})$.

#### 3.3.1  Relation with Sparse GPs using pseudo-inputs

The sparse GP using pseudo-inputs (SPGP) was introduced in [1] and was later renamed to Fully Independent Training Conditional (FITC) model to fit in the systematic framework of [10]. Since

the sparse model introduced in Section 3.2 also uses a fully independent training conditional, we will stick to the first name to avoid possible confusion.

IDGP innovation with respect to SPGP consists in letting the pseudo data set lie in a different domain. If we set $g_{\text{SPGP}}(\mathbf{x}, \mathbf{z}) \equiv \delta(\mathbf{x} - \mathbf{z})$ where $\delta(\cdot)$ is a Dirac delta, we force the pseudo data set to lie in the input domain. Thus there is no longer a transformed space and the original SPGP model is retrieved. In this setting, the inducing features of IDGP play the role of SPGP's pseudo-inputs.

### 3.3.2 Relation with Sparse Multiscale GPs

Sparse Multiscale GPs (SMGPs) are presented in [11]. Seeking to generalise the SPGP model with ARD SE covariance function, they propose to use a different set of length-scales for each basis function. The resulting model presents a defective variance that is healed by adding heteroscedastic white noise. SMGPs, including the variance improvement, can be derived in a principled way as IDGPs:

$$g_{\text{SMGP}}(\mathbf{x}, \mathbf{z}) \equiv \frac{1}{\prod_{d=1}^{D} \sqrt{2\pi(c_d^2 - \ell_d^2)}} \exp\left[-\sum_{d=1}^{D} \frac{(x_d - \mu_d)^2}{2(c_d^2 - \ell_d^2)}\right] \quad \text{with } \mathbf{z} = \begin{bmatrix} \boldsymbol{\mu} \\ \mathbf{c} \end{bmatrix} \quad (9)$$

$$k_{\text{SMGP}}(\mathbf{x}, \mathbf{z}') = \exp\left[-\sum_{d=1}^{D} \frac{(x_d - \mu_d')^2}{2c_d'^2}\right] \prod_{d=1}^{D} \sqrt{\frac{\ell_d^2}{c_d'^2}} \quad (10)$$

$$k_{\text{SMGP}}(\mathbf{z}, \mathbf{z}') = \exp\left[-\sum_{d=1}^{D} \frac{(\mu_d - \mu_d')^2}{2(c_d^2 + c_d'^2 - \ell_d^2)}\right] \prod_{d=1}^{D} \sqrt{\frac{\ell_d^2}{c_d^2 + c_d'^2 - \ell_d^2}}. \quad (11)$$

With this approximation, each basis function has its own centre $\boldsymbol{\mu} = [\mu_1, \ \mu_2, \ldots, \ \mu_d]^\top$ and its own length-scales $\mathbf{c} = [c_1, \ c_2, \ldots, \ c_d]^\top$, whereas global length-scales $\{\ell_d\}_{d=1}^{D}$ are shared by all inducing features. Equations (10) and (11) are derived from (4) and (5) using (1) and (9). The integrals defining $k_{\text{SMGP}}(\cdot, \cdot)$ converge if and only if $c_d^2 \geq \ell_d^2, \forall_d$, which suggests that other values, even if permitted in [11], should be avoided for the model to remain well defined.

### 3.3.3 Frequency Inducing Features GP

If the target function can be described more compactly in the frequency domain than in the input domain, it can be advantageous to let the pseudo data set lie in the former domain. We will pursue that possibility for the case where the input domain covariance is the ARD SE. We will call the resulting sparse model Frequency Inducing Features GP (FIFGP).

Directly applying the Fourier transform is not possible because the target function is not square integrable (it has constant power $\sigma_0^2$ everywhere, so (5) does not converge). We will workaround this by windowing the target function in the region of interest. It is possible to use a square window, but this results in the covariance being defined in terms of the complex error function, which is very slow to evaluate. Instead, we will use a Gaussian window[3]. Since multiplying by a Gaussian in the input domain is equivalent to convolving with a Gaussian in the frequency domain, we will be working with a blurred version of the frequency space. This model is defined by:

$$g_{\text{FIF}}(\mathbf{x}, \mathbf{z}) \equiv \frac{1}{\prod_{d=1}^{D} \sqrt{2\pi c_d^2}} \exp\left[-\sum_{d=1}^{D} \frac{x_d^2}{2c_d^2}\right] \cos\left(\omega_0 + \sum_{d=1}^{D} x_d \omega_d\right) \quad \text{with } \mathbf{z} = \boldsymbol{\omega} \quad (12)$$

$$k_{\text{FIF}}(\mathbf{x}, \mathbf{z}') = \exp\left[-\sum_{d=1}^{D} \frac{x_d^2 + c_d^2 \omega_d'^2}{2(c_d^2 + \ell_d^2)}\right] \cos\left(\omega_0' + \sum_{d=1}^{D} \frac{c_d^2 \omega_d' x_d}{c_d^2 + \ell_d^2}\right) \prod_{d=1}^{D} \sqrt{\frac{\ell_d^2}{c_d^2 + \ell_d^2}} \quad (13)$$

$$k_{\text{FIF}}(\mathbf{z}, \mathbf{z}') = \exp\left[-\sum_{d=1}^{D} \frac{c_d^2(\omega_d^2 + \omega_d'^2)}{2(2c_d^2 + \ell_d^2)}\right] \left(\exp\left[-\sum_{d=1}^{D} \frac{c_d^4(\omega_d - \omega_d')^2}{2(2c_d^2 + \ell_d^2)}\right] \cos(\omega_0 - \omega_0')\right.$$

$$\left. + \exp\left[-\sum_{d=1}^{D} \frac{c_d^4(\omega_d + \omega_d')^2}{2(2c_d^2 + \ell_d^2)}\right] \cos(\omega_0 + \omega_0')\right) \prod_{d=1}^{D} \sqrt{\frac{\ell_d^2}{2c_d^2 + \ell_d^2}}. \quad (14)$$

The inducing features are $\boldsymbol{\omega} = [\omega_0, \omega_1, \ldots, \omega_d]^\top$, where $\omega_0$ is the phase and the remaining components are frequencies along each dimension. In this model, both global length-scales $\{\ell_d\}_{d=1}^D$ and window length-scales $\{c_d\}_{d=1}^D$ are shared, thus $c_d' = c_d$. Instances (13) and (14) are induced by (12) using (4) and (5).

### 3.3.4 Time-Frequency Inducing Features GP

Instead of using a single window to select the region of interest, it is possible to use a different window for each feature. We will use windows of the same size but different centres. The resulting model combines SPGP and FIFGP, so we will call it Time-Frequency Inducing Features GP (TFIFGP). It is defined by $g_{\mathrm{TFIF}}(\mathbf{x}, \mathbf{z}) \equiv g_{\mathrm{FIF}}(\mathbf{x} - \boldsymbol{\mu}, \boldsymbol{\omega})$, with $\mathbf{z} = \begin{bmatrix} \boldsymbol{\mu}^\top & \boldsymbol{\omega}^\top \end{bmatrix}^\top$. The implied inter-domain and transformed-domain instances of the covariance function are:

$$k_{\mathrm{TFIF}}(\mathbf{x}, \mathbf{z}') = k_{\mathrm{FIF}}(\mathbf{x} - \boldsymbol{\mu}', \boldsymbol{\omega}'), \qquad k_{\mathrm{TFIF}}(\mathbf{z}, \mathbf{z}') = k_{\mathrm{FIF}}(\mathbf{z}, \mathbf{z}') \exp\left[ -\sum_{d=1}^D \frac{(\mu_d - \mu_d')^2}{2(2c_d^2 + \ell_d^2)} \right]$$

FIFGP is trivially obtained by setting every centre to zero $\{\boldsymbol{\mu}_i = \mathbf{0}\}_{i=1}^m$, whereas SPGP is obtained by setting window length-scales $\mathbf{c}$, frequencies and phases $\{\boldsymbol{\omega}_i\}_{i=1}^m$ to zero. If the window length-scales were individually adjusted, SMGP would be obtained.

While FIFGP has the modelling power of both FIFGP and SPGP, it might perform worse in practice due to it having roughly twice as many hyperparameters, thus making the optimisation problem harder. The same problem also exists in SMGP. A possible workaround is to initialise the hyperparameters using a simpler model, as done in [11] for SMGP, though we will not do this here.

## 4 Experiments

In this section we will compare the proposed approximations FIFGP and TFIFGP with the current state of the art, SPGP on some large data sets, for the same number of inducing features/inputs and therefore, roughly equal computational cost. Additionally, we provide results using a full GP, which is expected to provide top performance (though requiring an impractically big amount of computation). In all cases, the (input-domain) covariance function is the ARD SE (1).

We use four large data sets: *Kin-40k*, *Pumadyn-32nm*[4] (describing the dynamics of a robot arm, used with SPGP in [1]), *Elevators* and *Pole Telecomm*[5] (related to the control of the elevators of an F16 aircraft and a telecommunications problem, and used in [12, 13, 14]). Input dimensions that remained constant throughout the training set were removed. Input data was additionally centred for use with FIFGP (the remaining methods are translation invariant). *Pole Telecomm* outputs actually take discrete values in the 0-100 range, in multiples of 10. This was taken into account by using the corresponding quantization noise variance ($10^2/12$) as lower bound for the noise hyperparameter[6].

Hyperparameters are initialised as follows: $\sigma_0^2 = \frac{1}{n} \sum_{j=1}^n y_j^2$, $\sigma^2 = \sigma_0^2/4$, $\{\ell_d\}_{d=1}^D$ to one half of the range spanned by training data along each dimension. For SPGP, pseudo-inputs are initialised to a random subset of the training data, for FIFGP window size $\mathbf{c}$ is initialised to the standard deviation of input data, frequencies are randomly chosen from a zero-mean $\ell_d^{-2}$-variance Gaussian distribution, and phases are obtained from a uniform distribution in $[0 \ldots 2\pi)$. TFIFGP uses the same initialisation as FIFGP, with window centres set to zero. Final values are selected by evidence maximisation.

Denoting the output average over the training set as $\overline{y}$ and the predictive mean and variance for test sample $y_{*l}$ as $\mu_{*l}$ and $\sigma_{*l}$ respectively, we define the following quality measures: Normalized Mean Square Error (NMSE) $\langle (y_{*l} - \mu_{*l})^2 \rangle / \langle (y_{*l} - \overline{y})^2 \rangle$ and Mean Negative Log-Probability (MNLP) $\frac{1}{2} \langle (y_{*l} - \mu_{*l})^2 / \sigma_{*l}^2 + \log \sigma_{*l}^2 + \log 2\pi \rangle$, where $\langle \cdot \rangle$ averages over the test set.

For *Kin-40k* (Fig. 1, top), all three sparse methods perform similarly, though for high sparseness (the most useful case) FIFGP and TFIFGP are slightly superior. In *Pumadyn-32nm* (Fig. 1, bottom), only 4 out the 32 input dimensions are relevant to the regression task, so it can be used as an ARD capabilities test. We follow [1] and use a full GP on a small subset of the training data (1024 data points) to obtain the initial length-scales. This allows better minima to be found during optimisation. Though all methods are able to properly find a good solution, FIFGP and especially TFIFGP are better in the sparser regime. Roughly the same considerations can be made about *Pole Telecomm* and *Elevators* (Fig. 2), but in these data sets the superiority of FIFGP and TFIFGP is more dramatic.

Though not shown here, we have additionally tested these models on smaller, overfitting-prone data sets, and have found no noticeable overfitting even using $m > n$, despite the relatively high number of parameters being adjusted. This is in line with the results and discussion of [1].

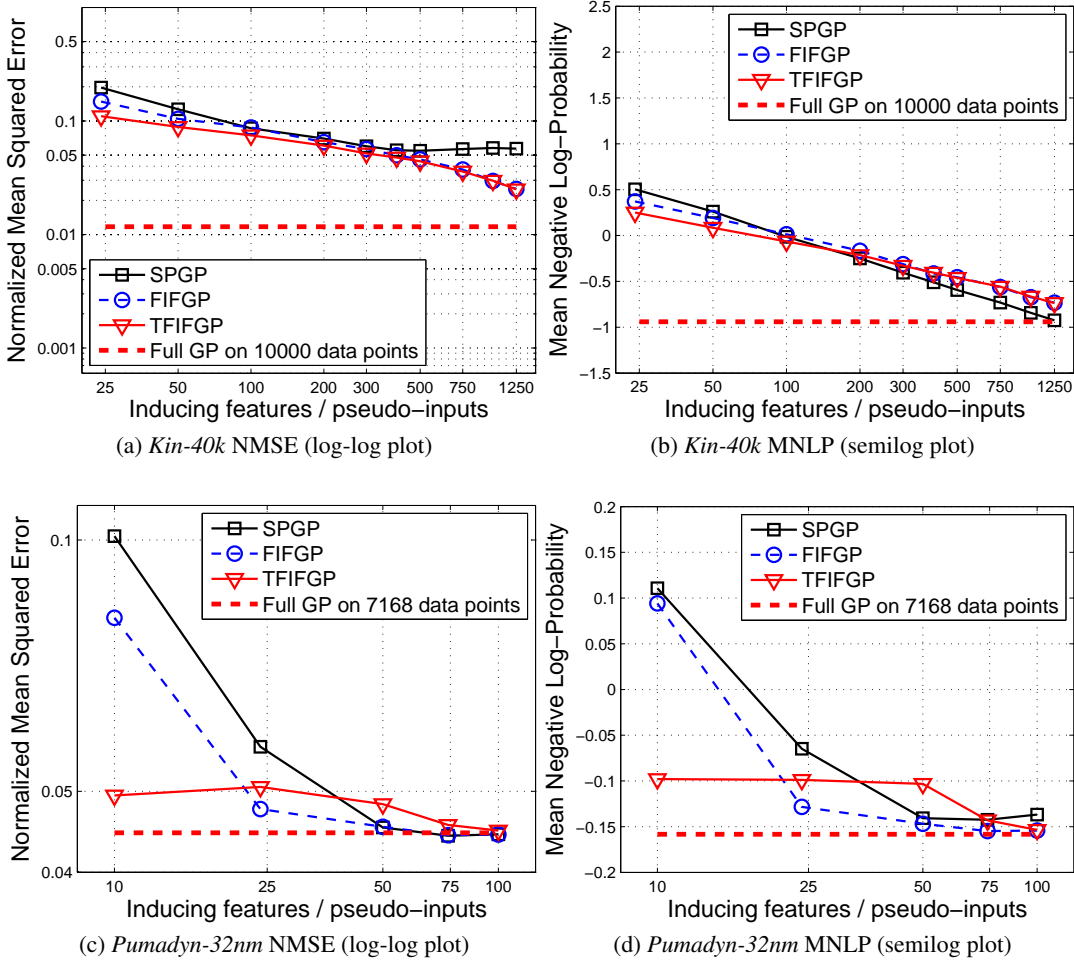

(a) *Kin-40k* NMSE (log-log plot)

(b) *Kin-40k* MNLP (semilog plot)

(c) *Pumadyn-32nm* NMSE (log-log plot)

(d) *Pumadyn-32nm* MNLP (semilog plot)

Figure 1: Performance of the compared methods on *Kin-40k* and *Pumadyn-32nm*.

## 5 Conclusions and extensions

In this work we have introduced IDGPs, which are able combine representations of a GP in different domains, and have used them to extend SPGP to handle inducing features lying in a different domain. This provides a general framework for sparse models, which are defined by a feature extraction function. Using this framework, SMGPs can be reinterpreted as fully principled models using a transformed space of local features, without any need for post-hoc variance improvements. Furthermore, it is possible to develop new sparse models of practical use, such as the proposed FIFGP and TFIFGP, which are able to outperform the state-of-the-art SPGP on some large data sets, especially for high sparsity regimes.

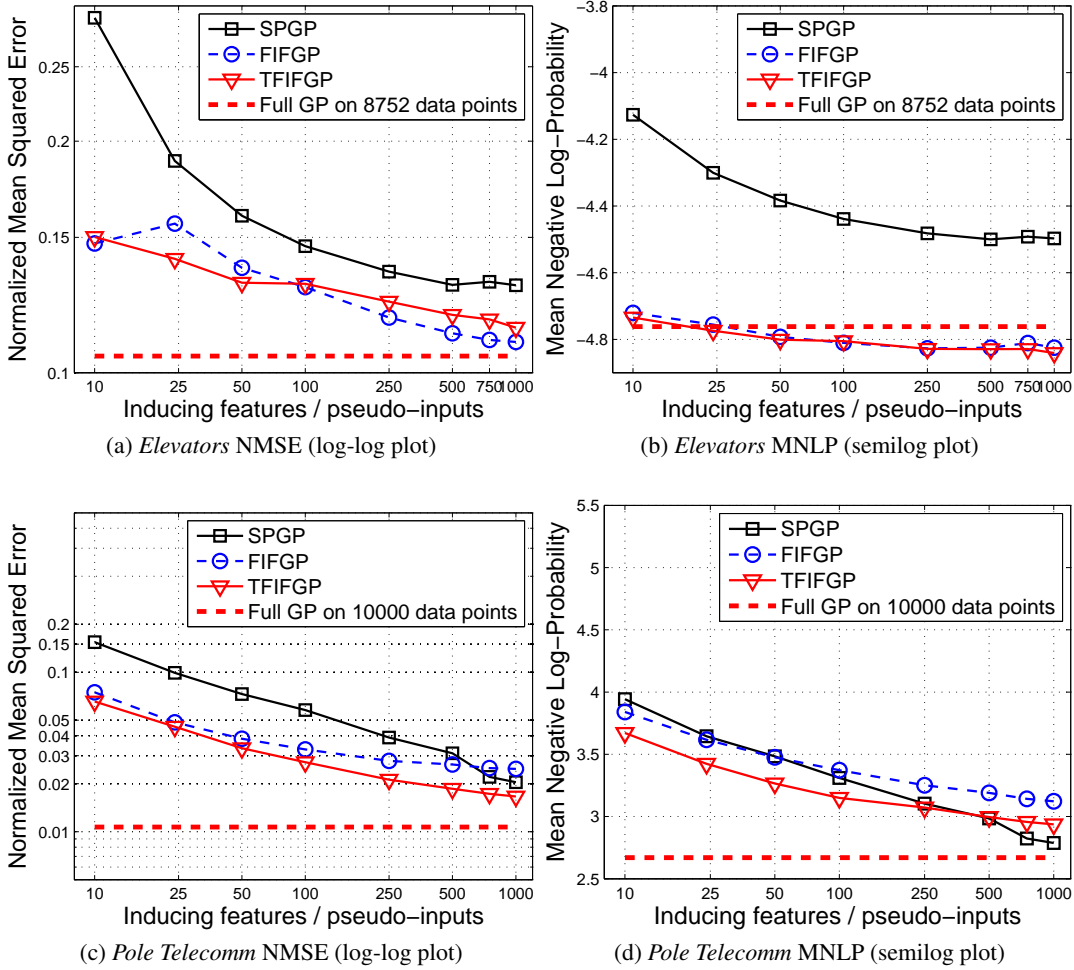

(a) *Elevators* NMSE (log-log plot)

(b) *Elevators* MNLP (semilog plot)

(c) *Pole Telecomm* NMSE (log-log plot)

(d) *Pole Telecomm* MNLP (semilog plot)

Figure 2: Performance of the compared methods on *Elevators* and *Pole Telecomm*.

Choosing a transformed space for the inducing features enables to use domains where the target function can be expressed more compactly, or where the evidence (which is a function of the features) is easier to optimise. This added flexibility translates as a detaching of the functional form of the input-domain covariance and the set of basis functions used to express the posterior mean.

IDGPs approximate full GPs optimally in the KL sense noted in Section 3.2, *for a given set of inducing features*. Using ML-II to select the inducing features means that models providing a good fit to data are given preference over models that might approximate the full GP more closely. This, though rarely, might lead to harmful overfitting. To more faithfully approximate the full GP and avoid overfitting altogether, our proposal can be combined with the variational approach from [15], in which the inducing features would be regarded as variational parameters. This would result in more constrained models, which would be closer to the full GP but might show reduced performance.

We have explored the case of regression with Gaussian noise, which is analytically tractable, but it is straightforward to apply the same model to other tasks such as robust regression or classification, using approximate inference (see [16]). Also, IDGPs as a general tool can be used for other purposes, such as modelling noise in the frequency domain, aggregating data from different domains or even imposing constraints on the target function.

### Acknowledgments

We would like to thank the anonymous referees for helpful comments and suggestions. This work has been partly supported by the Spanish government under grant TEC2008- 02473/TEC, and by the Madrid Community under grant S-505/TIC/0223.

## Footnotes

[1] We follow the common approach of subtracting the sample mean from the outputs and then assume a zero-mean model.

[2]Alternatively, (7) can be obtained by proposing a generic factorised form for the approximate conditional $p(\mathbf{f}|\mathbf{X}, \mathbf{Z}, \mathbf{u}) \approx q(\mathbf{f}|\mathbf{X}, \mathbf{Z}, \mathbf{u}) = \prod_{j=1}^n q_j(f_j|\mathbf{x}_j, \mathbf{Z}, \mathbf{u})$ and then choosing the set of functions $\{q_j(\cdot)\}_{j=1}^n$ so as to minimise the Kullback-Leibler (KL) divergence from the exact joint prior $\text{KL}(p(\mathbf{f}|\mathbf{X}, \mathbf{Z}, \mathbf{u}) p(\mathbf{u}|\mathbf{Z}) || q(\mathbf{f}|\mathbf{X}, \mathbf{Z}, \mathbf{u}) p(\mathbf{u}|\mathbf{Z}))$, as noted in [9], Section 2.3.6.

[3] A mixture of $m$ Gaussians could also be used as window without increasing the complexity order.

[4]*Kin-40k*: 8 input dimensions, 10000/30000 samples for train/test, *Pumadyn-32nm*: 32 input dimensions, 7168/1024 samples for train/test, using exactly the same preprocessing and train/test splits as [1, 3]. Note that their error measure is actually one half of the Normalized Mean Square Error defined here.

[5]*Pole Telecomm*: 26 non-constant input dimensions, 10000/5000 samples for train/test. *Elevators*: 17 non-constant input dimensions, 8752/7847 samples for train/test. Both have been downloaded from http://www.liaad.up.pt/~ltorgo/Regression/datasets.html

[6]If unconstrained, similar plots are obtained; in particular, no overfitting is observed.

## References

[1] E. Snelson and Z. Ghahramani. Sparse Gaussian processes using pseudo-inputs. In *Advances in Neural Information Processing Systems 18*, pages 1259–1266. MIT Press, 2006.

[2] A. J. Smola and P. Bartlett. Sparse greedy Gaussian process regression. In *Advances in Neural Information Processing Systems 13*, pages 619–625. MIT Press, 2001.

[3] M. Seeger, C. K. I. Williams, and N. D. Lawrence. Fast forward selection to speed up sparse Gaussian process regression. In *Proceedings of the 9th International Workshop on AI Stats*, 2003.

[4] V. Tresp. A Bayesian committee machine. *Neural Computation*, 12:2719–2741, 2000.

[5] L. Csató and M. Opper. Sparse online Gaussian processes. *Neural Computation*, 14(3):641–669, 2002.

[6] C. K. I. Williams and M. Seeger. Using the Nyström method to speed up kernel machines. In *Advances in Neural Information Processing Systems 13*, pages 682–688. MIT Press, 2001.

[7] C. E. Rasmussen and C. K. I. Williams. *Gaussian Processes for Machine Learning*. Adaptive Computation and Machine Learning. MIT Press, 2006.

[8] M. Alvarez and N. D. Lawrence. Sparse convolved Gaussian processes for multi-output regression. In *Advances in Neural Information Processing Systems 21*, pages 57–64, 2009.

[9] Ed. Snelson. *Flexible and efficient Gaussian process models for machine learning*. PhD thesis, University of Cambridge, 2007.

[10] J. Quiñonero-Candela and C. E. Rasmussen. A unifying view of sparse approximate Gaussian process regression. *Journal of Machine Learning Research*, 6:1939–1959, 2005.

[11] C. Walder, K. I. Kim, and B. Schölkopf. Sparse multiscale Gaussian process regression. In *25th International Conference on Machine Learning*. ACM Press, New York, 2008.

[12] G. Potgietera and A. P. Engelbrecht. Evolving model trees for mining data sets with continuous-valued classes. *Expert Systems with Applications*, 35:1513–1532, 2007.

[13] L. Torgo and J. Pinto da Costa. Clustered partial linear regression. In *Proceedings of the 11th European Conference on Machine Learning*, pages 426–436. Springer, 2000.

[14] G. Potgietera and A. P. Engelbrecht. Pairwise classification as an ensemble technique. In *Proceedings of the 13th European Conference on Machine Learning*, pages 97–110. Springer-Verlag, 2002.

[15] M. K. Titsias. Variational learning of inducing variables in sparse Gaussian processes. In *Proceedings of the 12th International Workshop on AI Stats*, 2009.

[16] A. Naish-Guzman and S. Holden. The generalized FITC approximation. In *Advances in Neural Information Processing Systems 20*, pages 1057–1064. MIT Press, 2008.

